# The Effects of Circuit Integration on a Feature Map Vector Quantizer

Jim Mann
MIT Lincoln Laboratory
244 Wood St.
Lexington, MA 02173
email: mann@vlsi.ll.mit.edu

## ABSTRACT

The effects of parameter modifications imposed by hardware constraints on a self-organizing feature map algorithm were examined. Performance was measured by the error rate of a speech recognition system which included this algorithm as part of the front-end processing. System parameters which were varied included weight (connection strength) quantization, adaptation quantization, distance measures and circuit approximations which include device characteristics and process variability. Experiments using the TI isolated word database for 16 speakers demonstrated degradation in performance when weight quantization fell below 8 bits. The competitive nature of the algorithm relaxes constraints on uniformity and linearity which makes it an excellent candidate for a fully analog circuit implementation. Prototype circuits have been fabricated and characterized following the constraints established through the simulation efforts.

## 1   Introduction

The self-organizing feature map algorithm developed by Kohonen [Kohonen, 1988] readily lends itself to the task of vector quantization for use in such areas as speech recognition. However, in considering practical implementations, it is necessary to

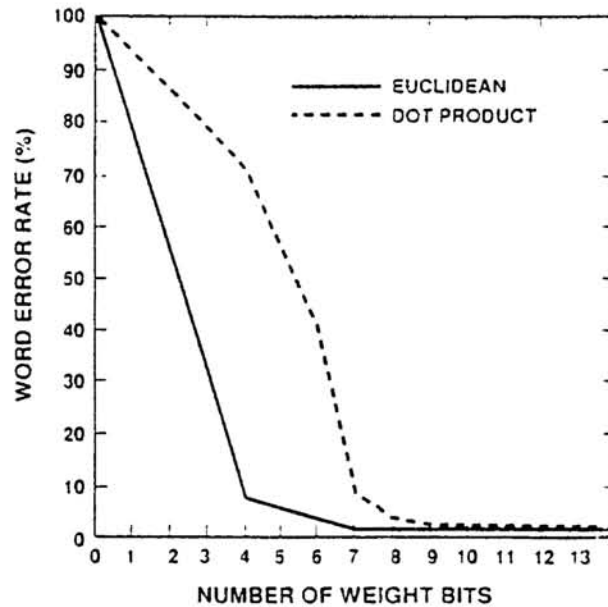

**Figure 1:** Recognition performance of the Euclidean and dot product activity calculators plotted as a function of weight precision.

understand the limitations imposed by circuitry on algorithm performance. In order to test the effects of these constraints on overall performance a simulation was written which permits ready variation of critical system parameters.

The feature map algorithm was placed in the frontend of a discrete hidden Markov model (HMM) speech recognition program as the vector quantizer (VQ) in order to track the effects of feature map algorithm modifications by monitoring overall word recognition accuracy. The system was tested on TI's 20 isolated word database consisting of 16 speakers. Each speaker had 1 training session consisting of 10 repetitions of each word in the vocabulary and 8 test sessions consisting of 2 repetitions of each word.

The key parameters tested include; quantization of both the weight coefficients and learning rule, and several different activation computations, the dot product and the mean squared error (i.e. squared Euclidean distance), as well as the circuit approximations to these calculators.

## 2   Results

A unique dependency between weight quantization and distance measure emerged from the simulations and is illustrated in the graph presented in Figure 1. The network equipped with the mean squared error activity calculator shows a "knee" in the word error rate at 6 bits of precision in the weight representation. The overall performance dropped only slightly between the essentially ideal floating point case, at 1.45% error rate, and the 6 bit case, at 2.99% error rate. At 4 bits, the error rate climbs to 7.62%. This still corresponds to a recognition accuracy of better than 92% but does show a marked degradation in performance.

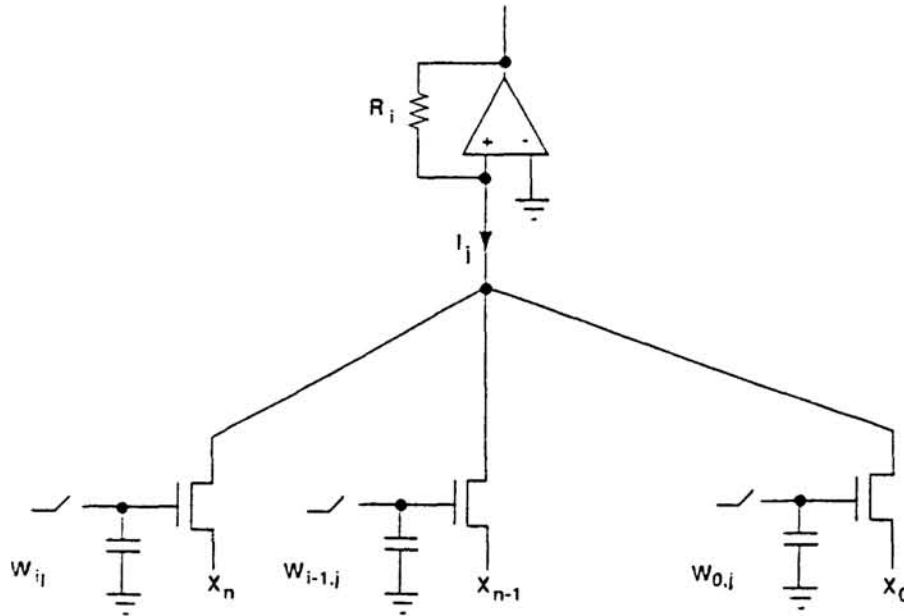

Figure 2: A circuit approximation to the dot product calculator.

The dot product does not degrade as gracefully with reduced precision in the weight representation as the mean squared error activity calculation. This is due to the normalization required on the input, and subsequently the weight vectors, which compresses the space onto the unit hypersphere. This step is necessary because of the inherent sensitivity of this metric to vector magnitude in making decisions of relative distance. Here the "knee" in the error curve occurs at 8 bits. Below 8 bits, performance drops off dramatically, reaching 40.6% error rate at 6 bits. The double precision floating point case starts off at 1.68% and is 3.44% at 8 bits.

Circuit approximations to these activity calculators were also included in the simulations. An approximation to the dot product operation can be implemented with single transistors operating in the ohmic region at each connection as illustrated in Figure 2.

These area related considerations can often overshadow the performance penalties associated with their implementation. The simulation results from this circuit approximation match the performance of the digital calculation of the dot product almost exactly as seen in Figure 3. This indicates that the performance of the system depends more on the monotonicity of the product operation performed at each connection then its linearity.

Effects of process variations on transistor thresholds were also examined. There appears to be a gradual decrease in system performance with increasing variability in transistor thresholds as seen in Figure 4. The cause of this phenomena remains to be investigated.

A weight adjustment rule which simplifies circuitry consists of quantizing the learning rate gain term. An integer step is added to or subtracted from the weight depending on the magnitude of the difference between it and the input. In the

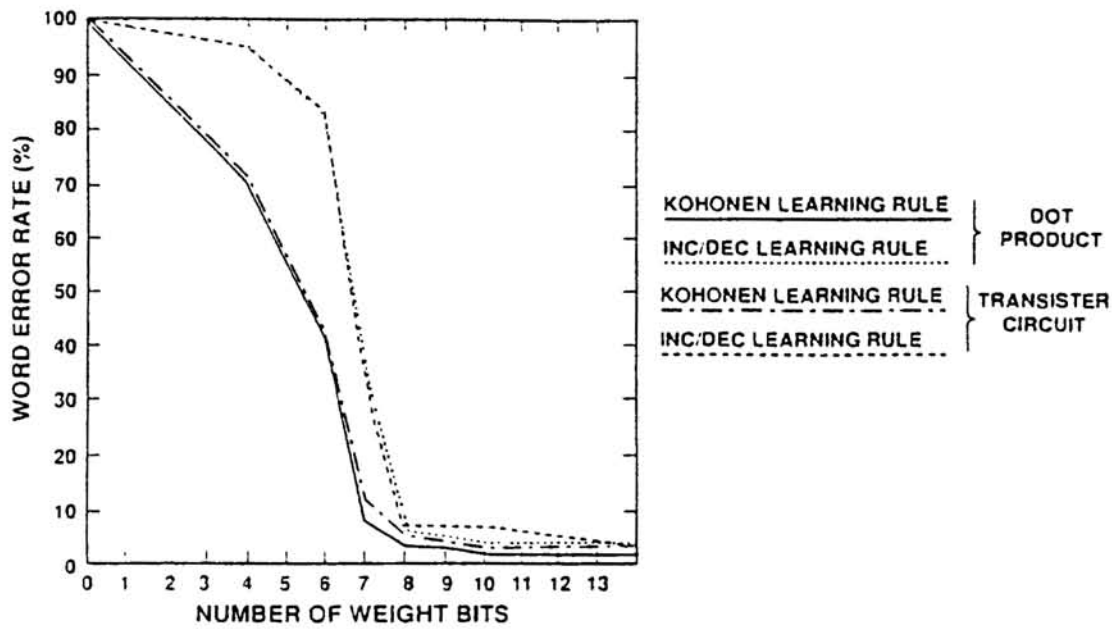

**Figure 3:** Similarity between the transistor circuit simulation and the digital calculation of the dot product

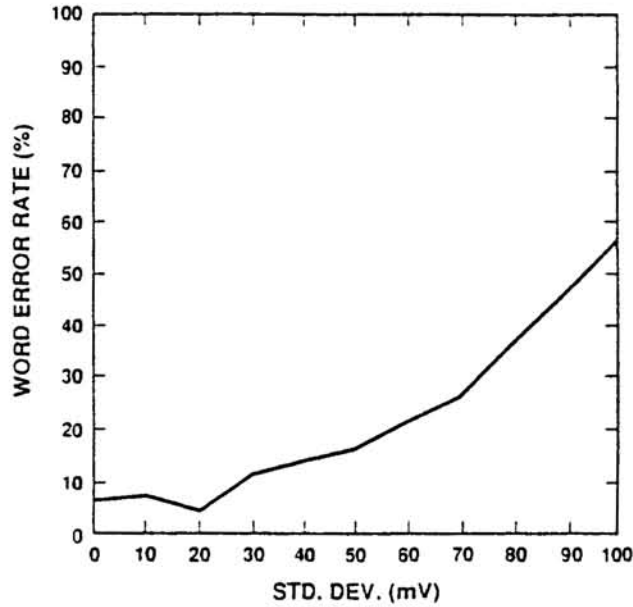

**Figure 4:** The effects of transistor threshold variation on recognition performance. (8 bit weight; Gaussian distributed, mean(Vth) = 0.75 volts).

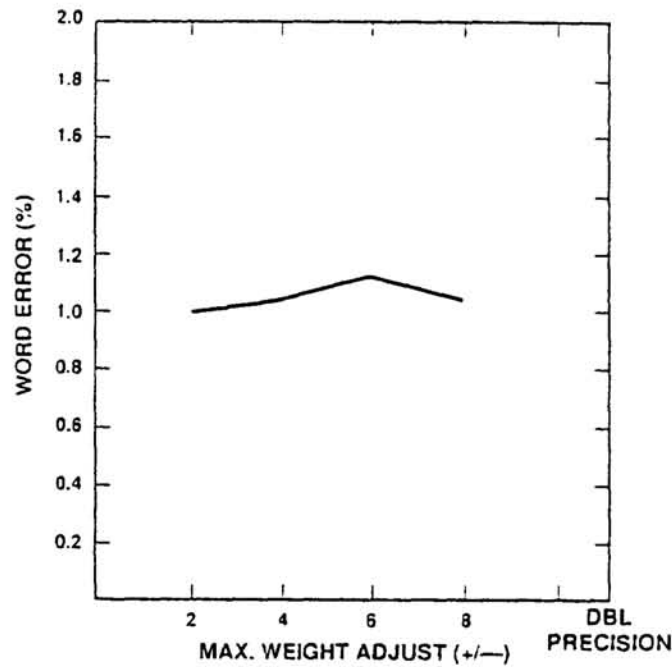

**Figure 5:** Word recognition error rate as a function of learning rate gain quantization.

simplest case. a fixed increment or decrement operation is performed based only upon the sign of the difference between the two terms. Even in this simplest case no degradation in performance was noted while using an 8 bit weight representation as demonstrated in the graph shown in Figure 5. In fact, performance was often improved over the original learning rule. The error rates using an increment/decrement learning rule with 8 weight bits was 0.97% and 2.0% for the mean squared error and the dot product, respectively.

An additional learning rule is being tested, targeted at a floating gate implementation which uses a "flash" EPROM memory structure at each synapse. Weight changes are restricted to positive adjustments locally while all negative adjustments are made globally to all weights. This corresponds to a forgetting term, or constant weight decay, in the learning rule. This rule was chosen to be compatible with one technique in non-volatile charge storage which allows selective write but only block erase.

## 3  Hardware

A prototype synaptic array and weight adaptation circuit have been designed and fabricated [Mann, 1989]. A single transistor synapse computes its contribution to the dot product activity calculation. The weight is stored dynamically as charge on the gate of the synapse transistor. The input is represented as a voltage on the drain of the transistor. The current through the transistor is proportional to the product of the gate voltage (i.e. the weight) and the drain voltage (i.e. the input strength) with the source connected to a virtual ground (see Figure 2). The sources of several of these synapse connected together form the accumulation needed to realize the dot product. Circuitry for accessing stored weight information has also been included.

The synapse array works as expected except for circuitry used to read the weight contents. This circuit requires very high on-chip voltages causing other circuits to latch-up when the clocks are turned on.

The weight adaptation circuit performs the simple increment/decrement operation based on the comparison between the input and weight magnitudes. Both quantities are first converted to a digital representation by a flash A/D converter before comparison. This circuit also performs the required refresh operation on weight contents, much like that required for dynamic RAM's but requiring analog charge storage. This insures that weight drift is constrained to lie within boundaries defined by the precision of the weight representation determined by the A/D conversion process. This circuit was functional in the refresh and increment modes, but would not decrement correctly.

Further tests are being conducted to establish the causes of the circuit problems detected thus far. Additional work is proceeding on a non-volatile charge storage version of this device. Some test structures have been fabricated and are currently being characterized for compatibility with this task.

This work was supported by the Department of the Air Force.

### References

T. Kohonen. (1988) Self-Organization and Associative Memory, Berlin: Springer-Verlag.

J. Mann & S. Gilbert. (1989) An Analog Self-Organizing Neural Network Chip. In D. S. Touretzky (ed.), *Advances in Neural Information Processing Systems 1*, 739-747. San Mateo, CA: Morgan Kaufmann.